# Self-organization of Hebbian Synapses
# in Hippocampal Neurons

Thomas H. Brown,[†] Zachary F. Mainen,[†] Anthony M. Zador,[†] and Brenda J. Claiborne[*]

[†] Department of Psychology
Yale University
New Haven, CT 06511

[*] Division of Life Sciences
University of Texas
San Antonio, TX 78285

## ABSTRACT

We are exploring the significance of biological complexity for neuronal computation. Here we demonstrate that Hebbian synapses in realistically-modeled hippocampal pyramidal cells may give rise to two novel forms of self-organization in response to structured synaptic input. First, on the basis of the electrotonic relationships between synaptic contacts, a cell may become tuned to a small subset of its input space. Second, the same mechanisms may produce clusters of potentiated synapses across the space of the dendrites. The latter type of self-organization may be functionally significant in the presence of nonlinear dendritic conductances.

## 1  INTRODUCTION

Long-term potentiation (LTP) is an experimentally observed form of synaptic plasticity that has been interpreted as an instance of a Hebbian modification (Kelso et al, 1986; Brown et al, 1990). The induction of LTP requires synchronous presynaptic activity and postsynaptic depolarization (Kelso et al, 1986). We have previously developed a detailed biophysical model of the LTP observed at synapses onto hippocampal region CA1 pyrami-

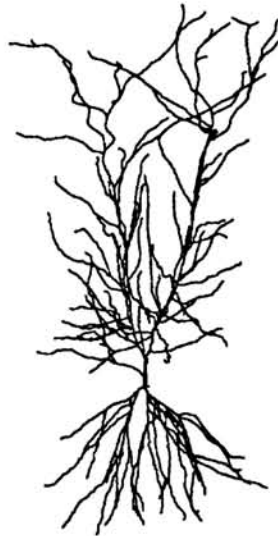

**Figure 1:** Two-dimensional projection of a reconstructed hippocampal CA1 pyramidal cell.

dal neurons (Zador et al, 1990). The synapses at which this form of LTP occurs are distributed across an extensive dendritic arbor (Fig. 1). During synaptic stimulation, the membrane voltage at each synapse is different. In this way, a biological neuron differs from the processing elements typically used in neural network models, where the postsynaptic activity can be represented by a single state variable. We have developed an electrotonic model based on an anatomically reconstructed neuron. We have used this model to explore how the spatial distribution of inputs and the temporal relationships of their activation affect synaptic potentiation.

## 2   THE NEURONAL MODEL

Standard compartmental modeling techniques were used to represent the electrical structure of hippocampal CA1 pyramidal cells.

### 2.1 MORPHOLOGY AND ELECTRICAL PARAMETERS

Morphometric data were obtained from three-dimensional reconstructions (Brown et al., 1991) of hippocampal neurons (Fig. 1). A correction factor was applied to the membrane area based on an estimate for spine density of $2 / \mu m$. The original measurements divided a single neuron into 3000-4000 cylinders with an average length of $5.5 \mu m$. For simulation purposes, this structure was collapsed into 300-400 compartments, preserving the connectivity pattern and changes in process diameter. Electrical constants were $R_m = 70 \, k\Omega\text{-}cm^2$, $C_m = 1 \, \mu F/cm^2$, $R_i = 200 \, \Omega\text{-}cm$ (Spruston & Johnston 1990). The membrane was electrically passive. Synaptic currents were modeled as the sum of fast AMPA and slow NMDA conductances on the head of a two-compartment spine (Zador et al., 1990). The AMPA conductance was represented by an alpha function (Jack et al., 1975) with time constant of $1.5 \, msec$ (Brown and Johnston, 1983). The NMDA conductance was represented by a more complicated function with two time constants and a voltage dependence due to voltage-sensitive channel blocking by $Mg^{2+}$ ions (see Zador et al., 1990; Brown et al. 1991). The initial peak conductances, $g_{AMPA}$ and $g_{NMDA}$, were set to 0.5 and 0.1 $nS$ respectively.

## 2.2 SIMULATION AND SYNAPTIC MODIFICATION

Simulations were run on a Sun 4/330 workstation using a customized version of NEURON, a simulator developed by Michael Hines (Hines, 1989). Prior to a simulation, 5 patterns of 40 synapses were selected at random from a pool of synapses distributed uniformly over the apical and basal dendrites. Simulations were divided into *trials* of 100 *msec*. At the beginning of each trial a particular pattern of synapses was activated synchronously (3 stimuli at intervals of 3 *msec*). The sequential presentation of all 5 selected patterns constituted an *epoch*. An entire simulation consisted of 20 presentation epochs. Over the course of each trial, membrane potential was computed at each location in the dendritic tree, and these voltages were used to compute weight changes $\Delta w_{ij}$ according to the Hebbian algorithm described below. After each trial, the actual peak AMPA conductances ($g_{AMPA}$, hereafter denoted $g_{syn}$) were scaled by the sigmoidal function

$$g_{syn} = \frac{g_{max}}{1 + e^{-\sigma(W_{ij} - 0.5g_{max})}} \tag{1}$$

where $\sigma$ determines the steepness of the sigmoid, and $g_{max}$ was set to 1.0 $nS$.

The rule for synaptic modification was based on a biophysical interpretation (Kairiss et al., 1991; Brown et al., 1991) of a generalized bilinear form of Hebbian algorithm (Brown et al., 1990):

$$\Delta w_{ij} = \alpha[a_i(t), a_j(t)] - \beta[a_i(t)] - \gamma[a_j(t)] - \delta, \tag{2}$$

where $\alpha$, $\beta$, and $\gamma$ are functionals, $\delta$ is a constant, $a_i(t)$ represents postsynaptic activity and $a_j(t)$ represents presynaptic activity. This equation specifies an interactive form of synaptic enhancement combined with three noninteractive forms of synaptic depression, all of which have possible neurobiological analogs (Brown et al, 1990). The interactive term was derived from a biophysical model of LTP induction in a spine (Zador et al., 1990). A simplified version of this model was used to compute the concentration of $Ca^{2+}$-bound calmodulin, [CaM-Ca$_4$]. It has been suggested that CaM-Ca$_4$ may trigger protein kinases responsible for LTP induction. In general [CaM-Ca$_4$] was a nonlinear function of subsynaptic voltage (Zador et al., 1990).

The biophysical mechanisms underlying synaptic depression are less well understood. The constant $\delta$ represents a passive decay process and was generally set to zero. The functional $\beta$ represents heterosynaptic depression based on postsynaptic activity. In these simulations, $\beta$ was proportional the amount of depolarization of the subsynaptic membrane from resting potential ($V_{syn} - V_{rest}$). The functional $\gamma$ represents homosynaptic depression based on presynaptic activity. Here, $\gamma$ was proportional to the AMPA conductance, which can be considered a measure of exclusively presynaptic activity because it is insensitive to postsynaptic voltage. The three activity-dependent terms were integrated over the period of the trial in order to obtain a measure of weight change. Reinterpreting $\alpha$, $\beta$, and $\gamma$ as constants, the equation is thus:

$$\Delta w_{ij} = \int_{trial} [\alpha[CamCa_4] - \beta(V_{syn} - V_{rest}) - \gamma g_{AMPA} - \delta] \, dt. \tag{3}$$

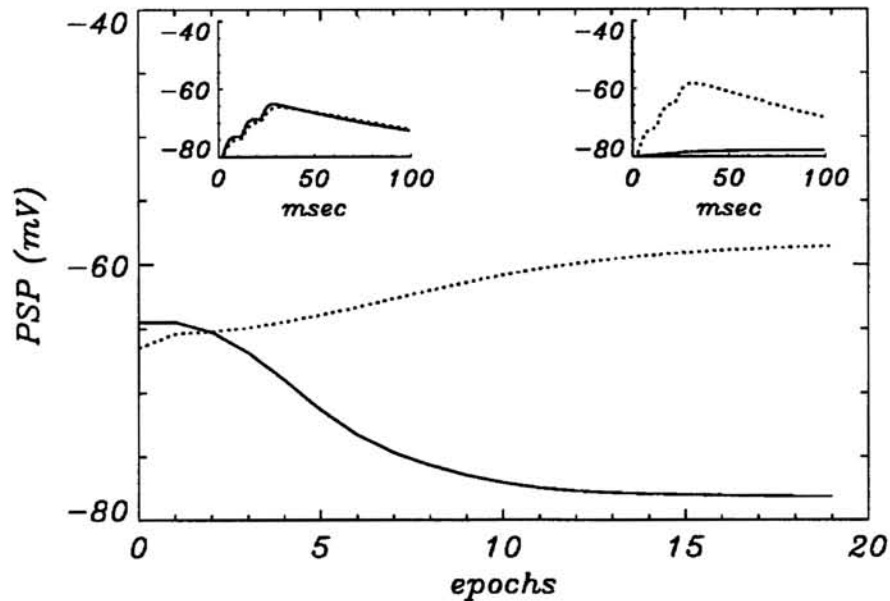

**Figure 2:** Interactions among Hebbian synapses produce differing global effects ("winning" and "losing" patterns) on the basis of the spatial distribution of synapses. The *PSP* (always measured at the soma) due to two different patterns of 40 synapses are plotted as a function of the presentation epoch. Initially, pattern 1 (*solid line*) evoked a slightly greater *PSP* than pattern 2 (*dotted line*; inset, top right). After 20 epochs these responses were reversed: the *PSP* due to pattern 1 was depressed while the *PSP* due to pattern 2 was potentiated (inset, top left).

## 3  RESULTS

Analysis of the simulations revealed self-organization in the form of differential modification of synaptic strengths (Mainen et al. 1990). Two aspects of the self-organization phenomena were distinguished. In some simulations, a form of *pattern selection* was observed in which clear "winners" and "losers" emerged. In other simulations, the average synaptic efficacy remained about the same, but spatial heterogeneities—*clustering*—of synaptic strength developed. Different measures were used to assess these phenomena.

### 3.1  PATTERN SELECTION

The change in the peak postsynaptic potential recorded at the soma (*PSP*) provided one useful measure of pattern selection. In many simulations, pattern selection resulted in a marked potentiation of the *PSP* due to some patterns and a depression of the *PSP* due to others. The *PSP* can be regarded as an indirect measure of the functional consequence of self-organization. In the simulation illustrated in Fig. 2, patterns of 40 synapses produced an average *PSP* of 15 *mV* before learning. After learning, responses ranged from 10% to 150% of this amount. Underlying pattern selection was a change in the average peak synaptic conductance for the pattern $\bar{g}_{syn}()$.[1] The initial value of $\bar{g}_{syn}$ was the same for all patterns, and its final value was bounded by eq. 1. In many simulations, $\bar{g}_{syn}$ approached the upper bound for some patterns and the lower bound for other patterns (Fig. 3). In this way, the neuron became selectively tuned to a subset of its original set of inputs. The specificity

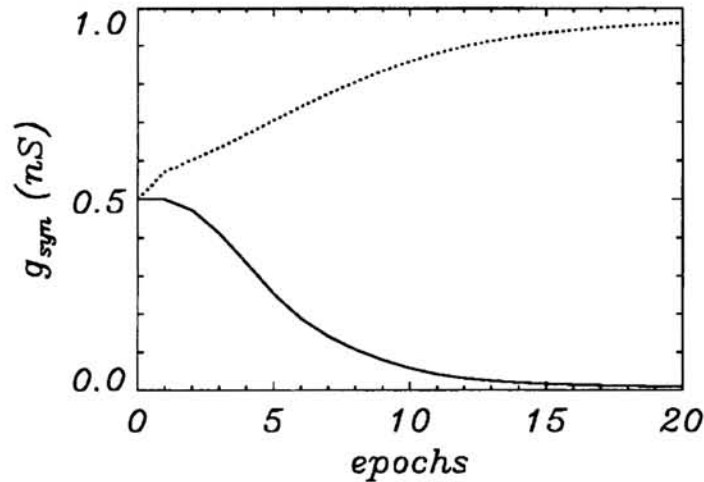

**Figure 3.** The mean synaptic conductance $\bar{g}_{syn}$ of two patterns is plotted as a function of the presentation epoch. Both patterns began with identical total synaptic strength (40 synapses with $g_{syn} = 0.5 \, nS$). Synaptic conductances were constrained to the range [0.0, 1.0] $nS$. After twenty epochs, $\bar{g}_{syn}$ of pattern 1 (*solid line*) approached the minimum of $0.0 \, nS$ while $\bar{g}_{syn}$ of pattern 2 (*dotted line*) approached the maximum of $1.0 \, nS$.

of this tuning was dependent on the parameter values of the neuronal model, learning rule, and stimulus set.

## 3.2 CLUSTER FORMATION

Heterogeneity in the spatial distribution of strengthened and weakened synapses was often observed. After learning, *spatial clusters* of synapses with similar conductances formed. These spatial heterogeneities can be illustrated in several ways. In one convenient method (see Brown et al., 1991), synapses are represented as colored points superimposed on a rendition of the neuronal morphology as illustrated in Fig. 1. By color-coding $g_{syn}$ for each synapse in a pattern, correlations in synaptic strength across dendritic space are immediately apparent. In a second method, better suited to the monochrome graphics available in the present text, the evolution of the variance of $g_{syn}$ is plotted as a function of time (Fig. 4). In the simulation illustrated here, the increase in variance was due to the formation of a single, relatively large cluster of strengthened synapses. Within other parameter regimes, multiple clusters of smaller size were formed.

## 4    DISCUSSION

The important differences between synaptic modifications in the biophysically-modeled neuron and those in simple processing elements arise from voltage gradients present in the realistic model (Brown et al., 1991; Kairiss et al., 1990). In standard processing elements,

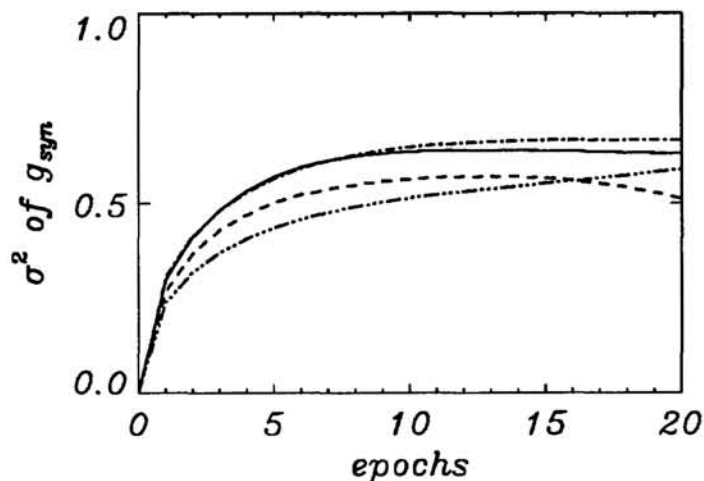

**Figure 4:** Synaptic heterogeneity is indicated by increases in the variance ($\sigma^2$) of the set of synaptic conductances for each pattern. The variances of the peak synaptic conductances ($g_{syn}$) of 4 patterns are plotted as a function of the epoch. The variance of all 4 patterns approached the theoretical maximum of $\sqrt{0.5}$. In this parameter regime, the variance was due to the potentiation of a single large cluster of synapes combined with the depression of other synapses.

a single state variable represents postsynaptic activity. In contrast, the critical subsynaptic voltages which represent postsynaptic activity in the neuron are correlated but are not strictly equal. The structure and electrical properties of the cell interact with its synaptic input to determine the precise spatiotemporal pattern of membrane voltage. Thus, the voltage at any synapse depends strongly on its electrotonic relationships to other active synapses. The way in which this local depolarization affects the nature of self-organization depends on the specific mechanisms of the synaptic modification rule. We have modeled a pair of opposing voltage-dependent mechanisms. An interactive potentiation mechanism (the functional $\alpha$) promotes cooperativity between spatially proximal synapses with temporally correlated activity. A heterosynaptic depression mechanism (the functional $\beta$), which is independent of presynaptic activity, promotes competition among spatially proximal synapses. Through mechanisms such as these, the specific electrotonic structure of a neuron predetermines a complex set of interactions between any given spatial distribution of synaptic inputs. We have shown that these higher-order interactions can give rise to self-organization with at least two interesting effects.

## 4.1 SPARSE REPRESENTATION

The phenomenon of pattern selection demonstrates how Hebbian self-organization may naturally tune neurons to respond to a subset of their input space. This tuning mechanism might allow a large field of neurons to develop a sparse coding of the activity in a set of input fibers, since each neuron would respond to a particular small portion of the input space. Sparse coding may be advantageous to associative learning and other types of neural computation (Kanerva, 1988).

## 4.2  CLUSTERING AND NONLINEAR COMPUTATION

The formation of clusters of strengthened synapses illustrates a property of Hebbian self-organization whose functional significance might only be appreciated in the presence of nonlinear (voltage-dependent) dendritic conductances. We have examined the self-organization process in an electrically passive neuron. Under these conditions, the presence of clustering within patterns has little effect on the observed output. In fact, it is known that hippocampal cells of the type modeled possess a variety of spatially heterogeneous nonlinear dendritic conductances (Jones et al., 1989). The computational role of such nonlinearities is just beginning to be explored. It is possible that interactions between synaptic clustering and nonlinear membrane patches may significantly affect both the performance of dendritic computations and the process of self-organization itself.

## Acknowledgments

This research was supported by grants from the Office of Naval Research, the Defense Advanced Research Projects Agency, and the Air Force Office of Scientific Research.

## Footnotes

[1] Although $\bar{g}_{syn}$ and the somatic *PSP* were generally correlated, the relationship between the two is not linear, as was often evident in simulations (compare initial trials in Figs. 2 and 3).

## References

Brown, T.H. and Johnston, D. (1983) Voltage-clamp analysis of mossy fiber synaptic input to hippocampal neurons. *J. Neurophysiol.* **50**: 487-507.

Brown, T.H., Kairiss, E.W. and Keenan, C.L. (1990) Hebbian synapses: biophysical mechanisms and algorithms. *Annu. Rev. Neurosci.* **13**: 475-512.

Brown, T.H., Zador, A.M., Mainen, Z.F. and Claiborne, B.J. (1991) Hebbian modifications in hippocampal neurons. In J. Davis and M. Baudry (eds.), *LTP: A Debate of Current Issues* (Cambridge, MA: MIT Press).

Hines, M. (1989) A program for simulation of nerve equations with branching geometries. *Int. J. Bio-Med Comp* **24**: 55-68.

Jack, J., Noble, A. and Tsien, R.W. (1975) *Electrical Current Flow in Excitable Membranes* (London: Oxford Univ. Press).

Jones, O.T., Kunze, D.L and Angelides, K.J. (1989) Localization and mobility of w-conotoxin-sensitive $Ca^{2+}$ channels in hippocampal CA1 neurons. *Science* **244**: 1189-1193.

Kairiss, E.W., Mainen, Z.F., Claiborne, B.J. and Brown, T.H. (1991) Dendritic control of hebbian compuations. In F. Eeckman (ed.), *Analysis and Modeling of Neural Systems* (Boston, MA: Kluwer Academic Publishers).

Kanerva, P. (1988) *Sparse distributed memory.* (Cambridge, MA: MIT Press).

Kelso, S.R., Ganong, Brown, T.H. (1986) Hebbian synapses in hippocampus. *Proc. Natl. Acad. Sci. USA* **83**: 5326-5330.

Mainen, Z.M., Zador, A.M., Claiborne, B. and Brown, T.H. (1990) Hebbian synapses induce feature mosaics in hippocampal dendrites. *Soc. Neurosci. Abstr.* **16**: 492.

Spruston, N. and Johnston, D. (1990) Whole-cell patch clamp analysis of the passive membrane properties of hippocampal neurons. *Soc. Neurosci. Abstr.* **16**: 1297.

Zador, A., Koch, C. and Brown, T.H. (1990) Biophysical model of a hebbian synapse. *Proc. Natl. Acad. Sci. USA* **87**: 6718-6722.